# Address Block Location with a Neural Net System

**Hans Peter Graf**                               **Eric Cosatto**

AT&T Bell Laboratories
Crawfords Corner Road
Holmdel, NJ 07733, USA

## Abstract

We developed a system for finding address blocks on mail pieces that can process four images per second. Besides locating the address block, our system also determines the writing style, handwritten or machine printed, and moreover, it measures the skew angle of the text lines and cleans noisy images. A layout analysis of all the elements present in the image is performed in order to distinguish drawings and dirt from text and to separate text of advertisement from that of the destination address.

A speed of more than four images per second is obtained on a modular hardware platform, containing a board with two of the NET32K neural net chips, a SPARC2 processor board, and a board with 2 digital signal processors. The system has been tested with more than 100,000 images. Its performance depends on the quality of the images, and lies between 85% correct location in very noisy images to over 98% in cleaner images.

## 1   INTRODUCTION

The system described here has been integrated into an address reading machine developed for the 'Remote Computer Reader' project of the United States Postal Service. While the actual reading of the text is done by other modules, this system solves one of the major problems, namely, finding reliably the location of the destination address. There are only a few constraints on how and where an address has to be written, hence they may appear in a wide variety of styles and layouts. Often an envelope contains advertising that includes images as well as text.

Sometimes, dirt covers part of the envelope image, including the destination address. Moreover, the image captured by the camera is thresholded and the reader is given a binary image. This binarization process introduces additional distortions; in particular, often the destination address is surrounded by a heavy texture. The high complexity of the images and their poor quality make it difficult to find the location of the destination address, requiring an analysis of all the elements present in the image. Such an analysis is compute-intensive and in our system it turned out to be the major bottleneck for a fast throughput. In fact, finding the address requires much more computation than reading it. Special-purpose hardware in the form of the NET32K neural net chips (Graf, Henderson, 90) is used to solve the address location problem.

Finding address blocks has been the focus of intensive research recently, as several companies are developing address reading machines (United States Postal Service 92). The wide variety of images that have to be handled has led other researchers to apply several different analysis techniques to each image and then try to combine the results at the end, see e.g. (Palumbo et al. 92). In order to achieve the throughput required in an industrial application, special purpose processors for finding connected components and/or for executing Hough transforms have been applied.

In our system we use the NET32K processor to extract geometrical features from an image. The high compute power of this chip allows the extraction of a large number of features simultaneously. From this feature representation, an interpretation of the image's content can then be achieved with a standard processor. Compared to an analysis of the original image, the analysis of the feature maps requires several orders of magnitude less computation. Moreover, the feature representation introduces a high level of robustness against noise. This paper gives a brief overview of the hardware platform in section 2 and then describes the algorithms to find the address blocks in section 3.

## 2 THE HARDWARE

The NET32K system has been designed to serve as a high-speed image processing platform, where neural nets as well as conventional algorithms can be executed. Three boards form the whole system. Two NET32K neural net chips are integrated with a sequencer and data formatting circuits on one board. The second board contains two digital signal processors (DSPs), together with 6 Mbytes of memory. Control of the whole system is provided by a board containing a SPARC2 processor plus 64 Mbytes of memory. A schematic of this system is shown in Figure 1.

Image buffering and communication with other modules in the address reader are handled by the board with the SPARC2 processor. When an image is received, it is sent to the DSP board and from there over to the NET32K processor. The feature maps produced by the NET32K processor are stored on the DSP board, while the SPARC2 starts with the analysis of the feature maps. The DSP's main task is formatting of the data, while the NET32K processor extracts all the features. Its speed of computation is more than 100 billion multiply-accumulates per second with operands that have one or two bits of resolution. Images with a size of 512x512 pixels are processed at a rate of more than 10 frames per second, and 64 convolution kernels, each with a size of 16x16 pixels, can be scanned simultaneously over the image. Each such kernel is tuned to detect the presence of a feature, such as a line, an edge or a corner.

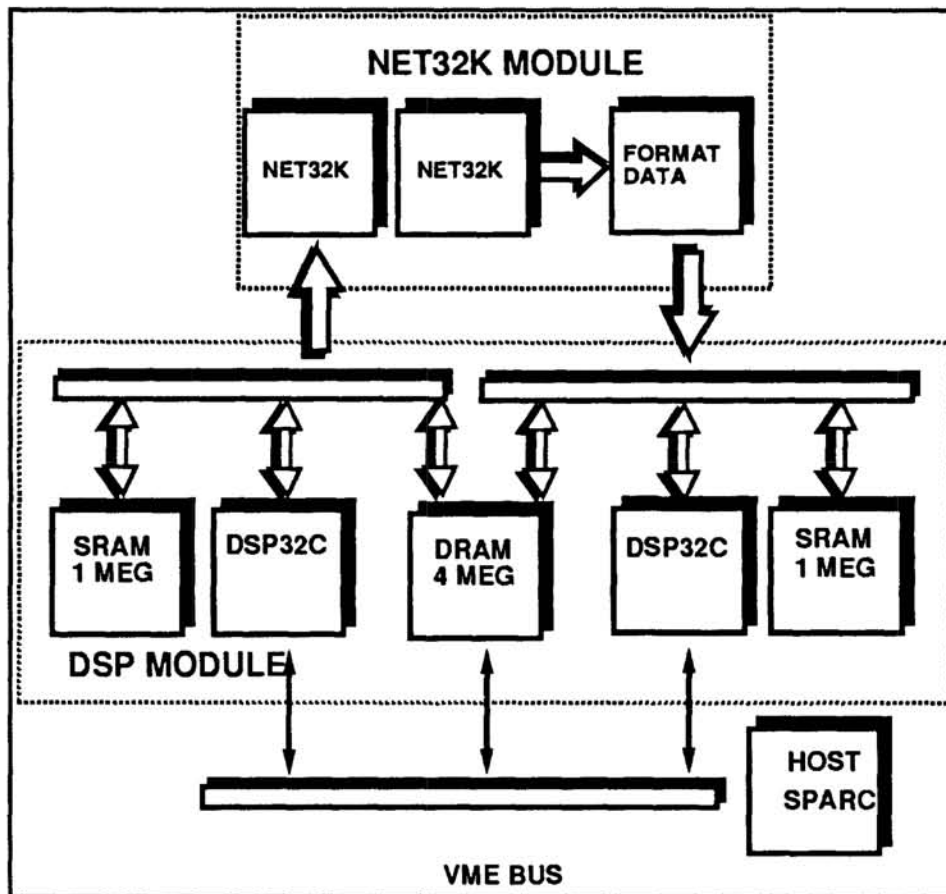

**Figure 1**: Schematic of the whole NET32K system. Each of the dashed boxes represents one 6U VME board. The arrows show the communication paths.

## 3. SEQUENCE OF ALGORITHMS

The final result of the address block location system is a box describing a tight bound around the destination address, if the address is machine printed. Of handwritten addresses, only the zip code is read, and hence, one has to find a tight boundary around the zip code. This information is then passed along to reader modules of the address reading machine. There is no a priori knowledge about the writing style. Therefore the system first has to discriminate between handwritten and machine printed text. At the end of the address block location process, additional algorithms are executed to improve the accuracy of the reader. An overview of the sequence of algorithms used to solve these tasks is shown in Figure 2. The whole process is divided into three major steps: Preprocessing, feature extraction, and high-level analysis based on the feature information.

### 3.1. Preprocessing

To quickly get an idea about the complexity of the image, a coarse evaluation of its layout is done. By sampling the density of the black pixels in various places of the image, one can see already whether the image is clean or noisy and whether the text is lightly printed or is dark.

The images are divided into four categories, depending on their darkness and the level of noise. This information is used in the subsequent processing to guide the choice of the features. Only about one percent of the pixels are taken into account for this analysis, therefore, it can be executed quickly on the SPARC2 processor.

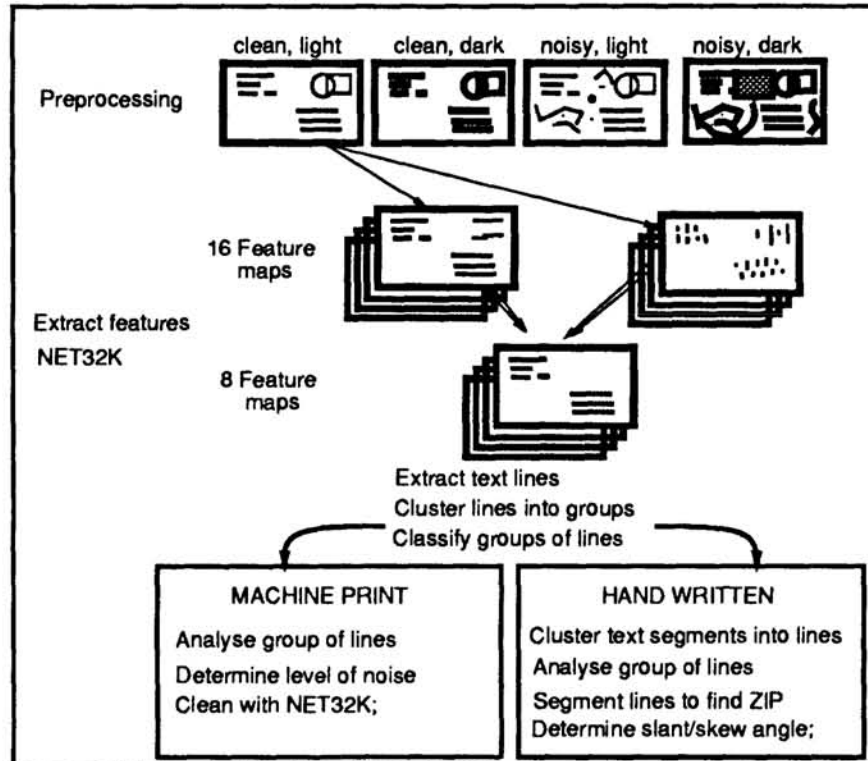

**Figure 2**: Schematic of the sequence of algorithms for finding the position of the address blocks.

## 3.2. Feature Extraction

After the preprocessing, the image is sent to the NET32K board where simple geometrical features, such as edges, corners and lines are extracted. Up to 16 different feature maps are generated, where a pixel in one of the maps indicates the presence of a feature in this location. Some of these feature maps are used by the host processor, for example, to decide whether text is handwritten or machine printed. Other feature maps are combined and sent once more through the NET32K processor in order to search for combinations of features representing more complex features. Typically, the feature maps are thresholded, so that only one bit per pixel is kept. More resolution of the computation results is available from the neural net chips, but in this way the amount of data that has to be analyzed is minimal, and one bit of resolution turned out to be sufficient.

Examples of kernels used for the detection of strokes and text lines are shown in Figure 3. In the chip, usually four line detectors of increasing height plus eight stroke detectors of different orientations are stored. Other detectors are tuned to edges and strokes of machine printed text. The line detectors respond to any black line of the proper height. Due to the large width of 16

pixels, a kernel stretches over one or even several characters. Hence a text line gives a response similar to that produced by a continuous black line. When the threshold is set properly, a text line in the original image produces a continuous line in the feature map, even across the gaps between characters and across small empty spaces between words. For an interpretation of a line feature map only the left and right end points of each connected component are stored. In this way one obtains a compact representation of the lines' positions that are well suited for the high-level analysis of the layout.

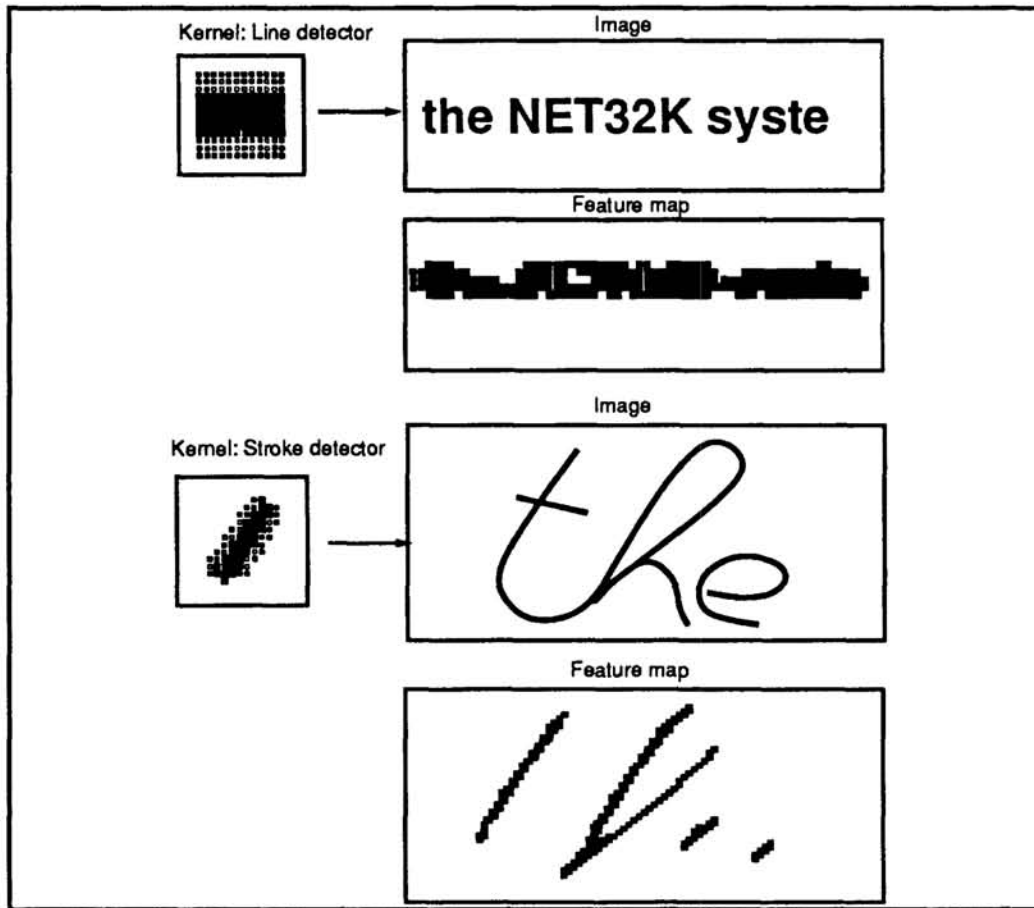

**Figure 3**:Examples of convolution kernels and their results. The kernels' sizes are 16x16 pixels, and their pixels' values are +1, 0, -1. The upper part illustrates the response of a line detector on a machine printed text line. The lower kernel extracts strokes of a certain orientation from handwritten text.

Handwritten lines are detected by a second technique, because they are more irregular in height and the characters may be spaced apart widely. Detectors for strokes, of the type shown in the lower half of Figure 3, are well suited for sensing the presence of handwritten text. The feature maps resulting form handwritten text tend to exhibit blobs of pixels along the text line. By smearing such feature maps in horizontal direction the responses of individual strokes are merged into lines that can then be used in the same way as described for the machine printed lines.

Horizontal smearing of text lines, combined with connected component analysis is a well-known

technique, often applied in layout analysis, to find words and whole lines of text. But when applied to the pixels of an image, such an approach works well only in clean images. As soon as there is noise present, this technique produces irregular responses. The key to success in a real world environment is robustness against noise. By extracting features first and then analyzing the feature maps, we drastically reduce the influence of noise. Each of the convolution kernels covers a range of 256 pixels and its response depends on several dozens of pixels inside this area. If pixels in the image are corrupted by noise, this has only a minor effect on the result of the convolution and, hence, the appearance of the feature map.

When the analysis is started, it is unknown, whether the address is machine printed or hand written. In order to distinguish between the two writing styles, a simple one-layer classifier looks at the results of four stroke detectors and of four line detectors. It can determine reliably whether text is handwritten or machine printed. Additional useful information that can be extracted easily from the feature maps, is the skew angle of handwritten text. People tend to write with a skew anywhere from -45 degrees to almost +90 degrees. In order to improve the accuracy of a reader, the text is first deskewed. The most time consuming part of this operation is to determine the skew angle of the writing. The stroke detector with the maximum response over a line is a good indicator of the skew angle of the text. We compared this simple technique with several alternatives and found it to be as reliable as the best other algorithm and much faster to compute.

## 3.3. High-level Analysis

The results of the feature extraction process are line segments, each one marked as handwritten or machine printed. Only the left and right end points of such lines are stored. At this point, there may still be line segments in this group that do not correspond to text, but rather to solid black lines or to line drawings. Therefore each line segment is checked, to determine whether the ratio of black and white pixels is that found typically in text.

Blocks of lines are identified by clustering the line segments into groups. Then each block is analyzed, to see whether it can represent the destination address. For this purpose such features as the number of lines in the block, its size, position, etc. are used. These features are entered into a classifier that ranks each of the blocks. Certain conditions, such as a size that is too large, or if there are too many text lines in the block, will lead to an attempt to split blocks. If no good result is obtained, clustering is tried again with a changed distance metric, where the horizontal and the vertical distances between lines are weighted differently.

If an address is machine printed, the whole address block is passed on to the reader, since not only the zip code, but the whole address, including the city name, the street name and the name of the recipient have to be read. A big problem for the reader present images of poor quality, particularly those with background noise and texture. State-of-the-art readers handle machine printed text reliably if the image quality is good, but they may fail totally if the text is buried in noise. For that reason, an address block is cleaned before sending it to the reader. Feature extraction with the NET32K board is used once more for this task, this time with detectors tuned to find all the strokes of the machine printed text. Applying stroke detectors with the proper width allows a good discrimination between the text and any noise. Even texture that consists of lines can be rejected reliably, if the line thickness of the texture is not the same as that of the text.

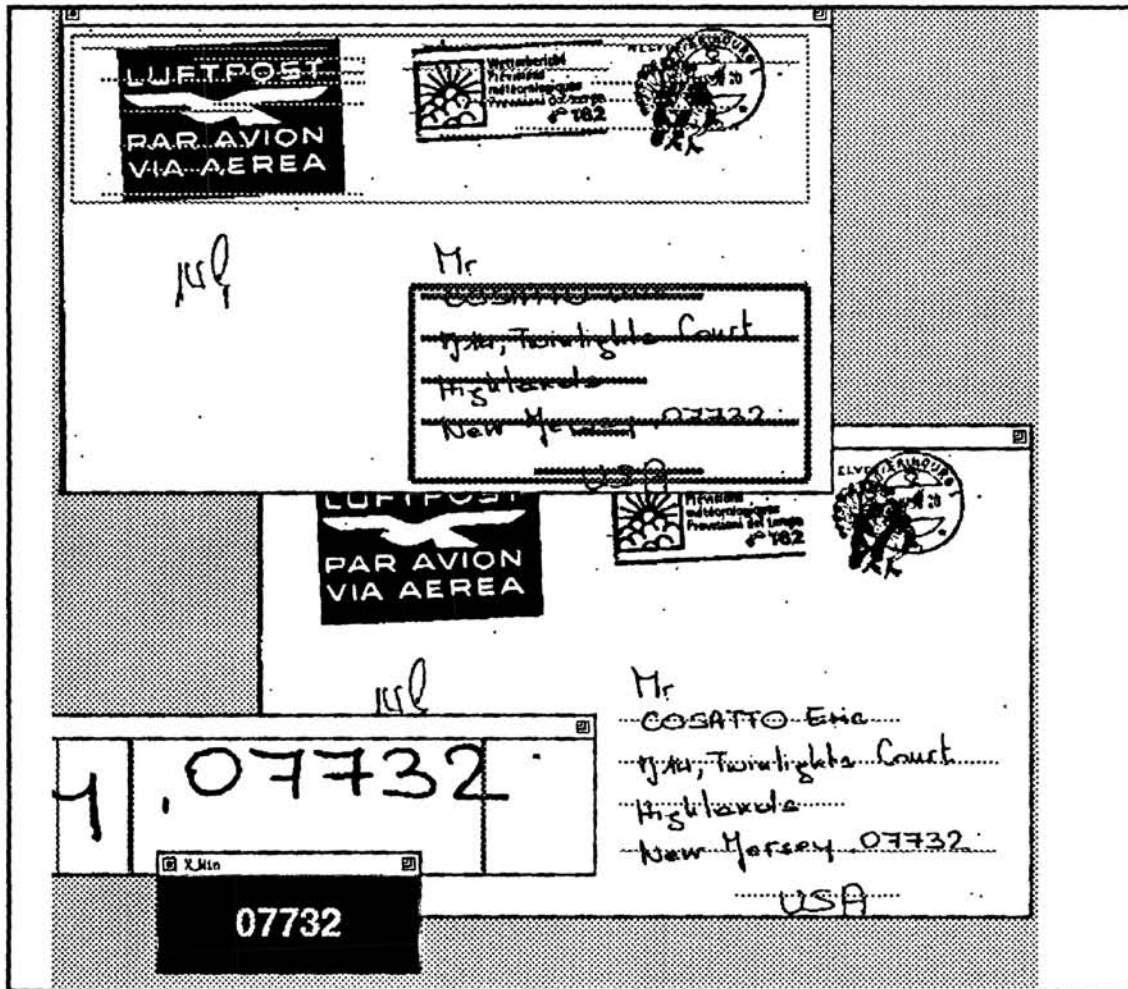

**Figure 4**: Example of an envelope image at various stages of the processing. Top: The result of the clustering process to find the bounding box of the address. Bottom right: The text lines within the address block are marked. Bottom left: Cuts in the text line with the zip code and below that the result of the reader. (The zip code is actually the second segment sent to the reader; the first one is the string 'USA').

If the address is handwritten, only the zip code is sent to the reader. In order to find the zip code, an analysis of the internal structure of the address block has to be done, which starts with finding the true text lines. Handwritten lines are often not straight, may be heavily skewed, and may contain large gaps. Hence simple techniques, such as connected component analysis, do not provide proper results. Clustering of the line segments obtained from the feature maps, provides a reliable solution of this problem. Once the lines are found, each one is segmented into words and some of them are selected as candidates for the zip code and are sent to the reader. Figure 4 shows an example of an envelope image as it progresses through the various processing steps.

The system has been tested extensively on overall more than 100,000 images. Most of these tests were done in the assembled address reader, but during development of the system, large

tests were also done with the address location module alone. One of the problems for evaluating the performance is the lack of an objective quality measure. When has an address been located correctly? Cutting off a small part of the address may not be detrimental to the final interpretation, while a bounding box that includes some additional text may slow the reader down too much, or it may throw off the interpretation. Therefore, it is not always clear when a bounding box, describing the address' location, is tight enough. Another important factor affecting the accuracy numbers is, how many candidate blocks one actually considers. For all these reasons, accuracy numbers given for address block location have to be taken with some caution. The results mentioned here were obtained by judging the images by eye. If images are clean and the address is surrounded by a white space larger than two line heights, the location is found correctly in more than 98% of the cases. Often more than one text block is found and of these the destination address is the first choice in 90% of the images, for a typical layout. If the image is very noisy, which actually happens surprisingly often, a tight bound around the address is found in 85% of the cases. These results were obtained with 5,000 images, chosen from more than 100,000 images to represent as much variety as possible. Of these 5,000 images more than 1,200 have a texture around the address, and often this texture is so dark that a human has difficulties to make out each character.

## 4. CONCLUSION

Most of our algorithms described here consist of two parts: feature extraction implemented with a convolution and interpretation, typically implemented with a small classifier. Surprisingly many algorithms can be cast into such a format. This common framework for algorithms has the advantage of facilitating the implementation, in particular when algorithms are mapped into hardware. Moreover, the feature extraction with large convolution kernels makes the system robust against noise. This robustness is probably the biggest advantage of our approach. Most existing automatic reading systems are very good as long as the images are clean, but they deteriorate rapidly with decreasing image quality.

The biggest drawback of convolutions is that they require a lot of computation. In fact, without special purpose hardware, convolutions are often too slow. Our system relies on the NET32K neural net chips to obtain the necessary throughput. The NET32K system is, we believe, at the moment the fastest board system for this type of computation. This speed is obtained by systematically exploiting the fact that only a low resolution of the computation is required. This allows to use analog computation inside the chip and hence much smaller circuits than would be the case in an all-digital circuit.

## References

United States Postal Service, (1992), Proc. Advanced Technology Conf., Vol. 3, Section on address block location: pp. 1221 - 1310.

P.W. Palumbo, S.N. Srihari, J. Soh, R. Sridhar, V. Demjanenko, (1992), "Postal Address Block Location in Real Time", IEEE COMPUTER, Vol. 25/7, pp. 34 - 42.

H.P. Graf and D. Henderson, (1990), "A Reconfigurable CMOS Neural Network", Digest IEEE Int. Solid State Circuits Conf. p. 144.
